# Designing Application-Specific
# Neural Networks
# Using the Genetic Algorithm

**Steven A. Harp, Tariq Samad, Aloke Guha**
Honeywell SSDC
1000 Boone Avenue North
Golden Valley, MN 55427

## ABSTRACT

We present a general and systematic method for neural network design based on the genetic algorithm. The technique works in conjunction with network learning rules, addressing aspects of the network's gross architecture, connectivity, and learning rule parameters. Networks can be optimised for various application-specific criteria, such as learning speed, generalisation, robustness and connectivity. The approach is model-independent. We describe a prototype system, *NeuroGENESYS*, that employs the backpropagation learning rule. Experiments on several small problems have been conducted. In each case, NeuroGENESYS has produced networks that perform significantly better than the randomly generated networks of its initial population. The computational feasibility of our approach is discussed.

## 1 INTRODUCTION

With the growing interest in the practical use of neural networks, addressing the problem of customising networks for specific applications is becoming increasingly critical. It has repeatedly been observed that different network structures and learning parameters can substantially affect performance. Such important aspects of neural network applications as generalisation, learning speed, connectivity and tolerance to network damage are strongly related to the choice of

network architecture. Yet there are few analytic results, and few heuristics, that can help the application developer design an appropriate network.

We have been investigating the use of the genetic algorithm (Goldberg, 1989; Holland, 1975) for designing application-specific neural networks (Harp, Samad and Guha, 1989ab). In our approach, the genetic algorithm is used to evolve appropriate network structures and values of learning parameters. In contrast, other recent applications of the genetic algorithm to neural networks (e.g., Davis [1988], Whitley [1988]) have largely restricted the role of the genetic algorithm to updating weights on a predetermined network structure—another logical approach.

Several first-generation neural network application development tools already exist. However, they are only partly effective: the complexity of the problem, our limited understanding of the interdependencies between various network design choices, and the extensive human effort involved permit only limited exploration of the design space. An objective of our research is the development of a next-generation neural network application development tool that can synthesise optimised custom networks. The genetic algorithm has been distinguished by its relative immunity to high dimensionality, local minima and noise, and it is therefore a logical candidate for solving the network optimisation problem.

## 2  GENETIC SYNTHESIS OF NEURAL NETWORKS

Fig. 1 outlines our approach. A network is represented by a *blueprint*—a bit-string that encodes a number of characteristics of the network, including structural properties and learning parameter values. Each blueprint directs the creation of an actual network with random initial weights. An instantiated network is trained using some predetermined training algorithm and training data, and the trained network can then be tested in various ways—e.g., on non-training inputs, after disabling some units, and after perturbing learned weight values. After testing, a network is evaluated—a *fitness* estimate is computed for it based on appropriate criteria. This process of instantiation, training, testing and evaluation is performed for each of a population of blueprints.

After the entire population is evaluated, the next *generation* of blueprints is produced. A number of *genetic operators* are employed, the most prominent of these being *crossover*, in which two parent blueprints are spliced together to produce a child blueprint (Goldberg, 1989). The higher the fitness of a blueprint, the greater the probability of it being selected as a parent for the subsequent generation. Characteristics that are found useful will thereby tend to be emphasized in the next generation, whereas harmful ones will tend to be suppressed.

The definition of network performance depends on the application. If the application requires good generalisation capabilities, the results of testing on (appropriately chosen) non-training data are important. If a network capable of real-time learning is required, the learning rate must be optimised. For fast response, the size of the network must be minimized. If hardware (especially VLSI) implementation is a consideration, low connectivity is essential. In most applications several such criteria must be considered. This important aspect of application-specific network design is covered by the fitness function. In our approach, the fitness of a network can be an arbitrary function of several distinct

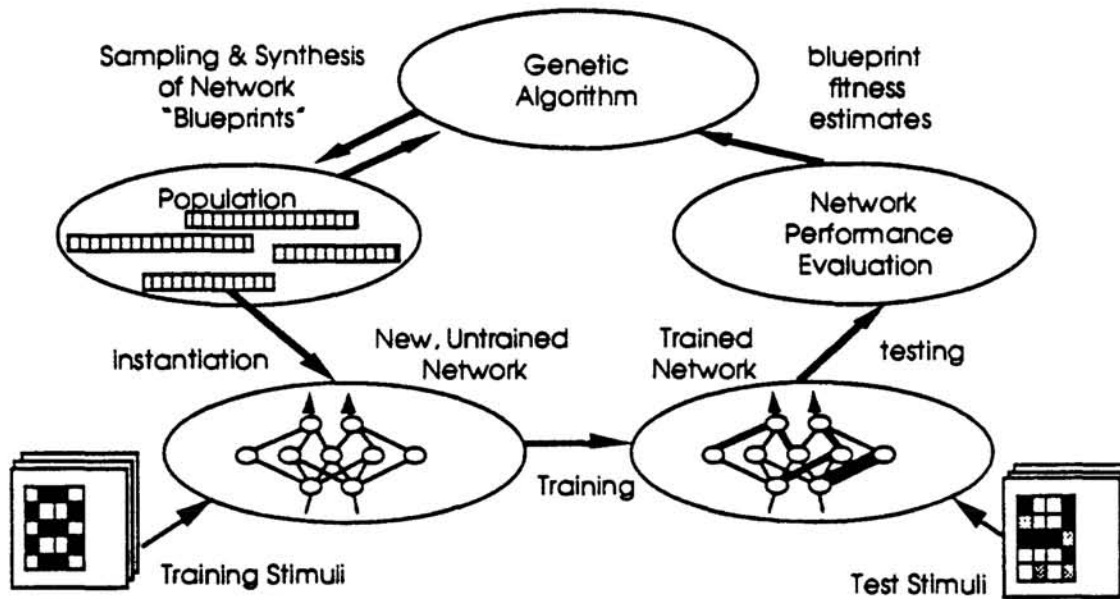

**Figure 1:** A population of network "blueprints" is cyclically updated by the genetic algorithm based on their fitness.

performance and cost criteria, some or all of which can thereby be simultaneously optimized.

## 3 NEUROGENESYS

Our approach is model-independent: it can be applied to any existing or future neural network model (including models without a training component). As a first prototype implementation we have developed a working system called *NeuroGENESYS*. The current implementation uses a variant (Samad, 1988) of the backpropagation learning algorithm (Werbos, 1974; Rumelhart, Hinton, and Williams, 1985) as the training component and is restricted to feedforward networks.

Within these constraints, NeuroGENESYS is a reasonably general system. Networks can have arbitrary directed acyclic graph structures, where each vertex of the graph corresponds to an *area* or layer of units and each edge to a *projection* from one area to another. Units in an area have a spatial organization; the current system arrays units in 2 dimensions. Each projection specifies independent radii of connectivity, one for each dimension. The radii of connectivity allow localized receptive field structures. Within the receptive fields connection densities can be specified. Two learning parameters are associated with both projections and areas. Each projection has a learning rate parameter ("$\eta$" in backpropagation) and a decay rate for $\eta$. Each area has $\eta$ and $\eta$-decay parameters for threshold weights.

These network characteristics are encoded in the genetic blueprint. This bitstring is composed of several segments, one for each area. An area segment consists of an area parameter specification (APS) and a variable number of projection

specification fields (PSFs), each of which describes a projection from the area to some other area. Both the APS and the PSF contain values for several parameters for areas and projections respectively. Fig. 2 shows a simple area segment. Note that the target of a projection can be specified through either *Absolute* or *Relative* addressing. More than one projections are possible between two given areas; this allows the generation of receptive field structures at different scales and with different connection densities, and it also allows the system to model the effect of larger initial weights. In our current implementation, all initial weights are randomly generated small values from a fixed uniform distribution. In the near future, we intend to incorporate some aspects of the distribution in the genetic blueprint.

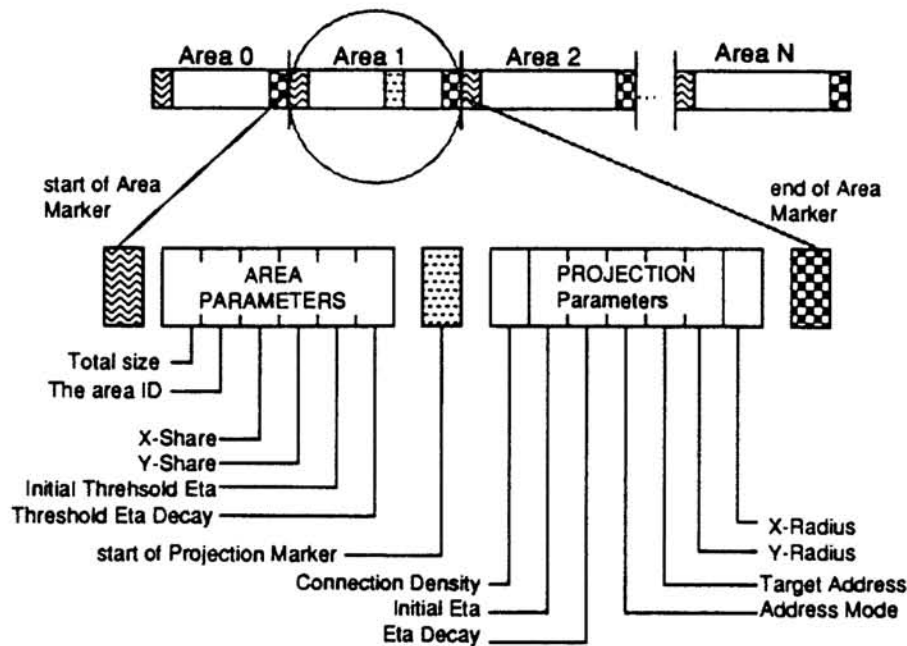

**Figure 2: Network Blueprint Representation**

In NeuroGENESYS, the score of a blueprint is computed as a linear weighted sum of several performance and cost criteria, including learning speed, the results of testing on a "test set", the numbers of units and weights in the network, the results of testing (on the training set) after disabling some of the units, the results of testing (on the training set) after perturbing the learned weight values, the average fanout of the network, and the maximum fanout for any unit in the network. Other criteria can be incorporated as needed. The user of Neuro-GENESYS supplies the weighting factors at the start of the experiment, thereby controlling which aspects of the network are to be optimized.

## 4   EXPERIMENTS

NeuroGENESYS can be used for both classification and function approximation problems. We have conducted experiments on three classification problems—digit recognition from 4×8 pixel images, exclusive-OR (XOR), and simple convexity

detection; and one function approximation problem—modeling one cycle of a sine function. Various combinations of the above criteria have been used. In most experiments NeuroGENESYS has produced appropriate network designs in a relatively small number of generations ($< 50$).

Our first experiment was with digit recognition, and NeuroGENESYS produced a solution that surprised us: The optimized networks had no hidden layers yet learned perfectly. It had not been obvious to us that this digit recognition problem is linearly separable. Even in the simple case of no-hidden-layer networks, our earlier remarks on application-specific design can be appreciated. When NeuroGENESYS was asked to optimize for average fanout for the digit recognition task as well as for perfect learning, the best network produced learned perfectly (although comparatively slowly) and had an average fanout of three connections per unit; with learning speed as the sole optimization criterion, the best network produced learned substantially faster (48 iterations) but it had an average fanout of almost an order of magnitude higher.

The XOR problem, of course, is prototypically non-linearly-separable. In this case, NeuroGENESYS produced many fast-learning networks that had a "bypass" connection from the input layer directly to the output layer (in addition to connections to and from hidden layers); it is an as yet unverified hypothesis that these bypass connections accelerate learning.

In one of our experiments on the sine function problem, NeuroGENESYS was asked to design networks for moderate accuracy—the error cutoff during training was relatively high. The networks produced typically had one hidden layer of two units, which is the minimum possible configuration for a sufficiently crude approximation. When the experiment was repeated with a low error cutoff, intricate multilayer structures were produced that were capable of modeling the training data very accurately (Fig. 3). Fig. 4 shows the learning curve for one sine function experiment. The "Average" and "Best" scores are over all individuals in the generation, while "Online" and "Offline" are running averages of Average and Best, respectively. Performance on this problem is quite sensitive to initial weight values, hence the non-monotonicity of the Best curve. Steady progress overall was still being observed when the experiment was terminated.

We have conducted control studies using random search (with best retention) instead of the genetic algorithm. The genetic algorithm has consistently proved superior. Random search is the weakest possible optimization procedure, but on the other hand there are few sophisticated alternatives for this problem—the search space is discontinuous, largely unknown, and highly nonlinear.

## 5 COMPUTATIONAL EFFICIENCY

Our approach requires the evaluation of a large number of networks. Even on some of our small-scale problems, experiments have taken a week or longer, the bottleneck being the neural network training algorithm. While computational feasibility is a real concern, for several reasons we are optimistic that this approach will be practical for realistic applications:

- The hardware platform for our experiments to date has been a Symbolics computer without any floating-point support. This choice has been ideal

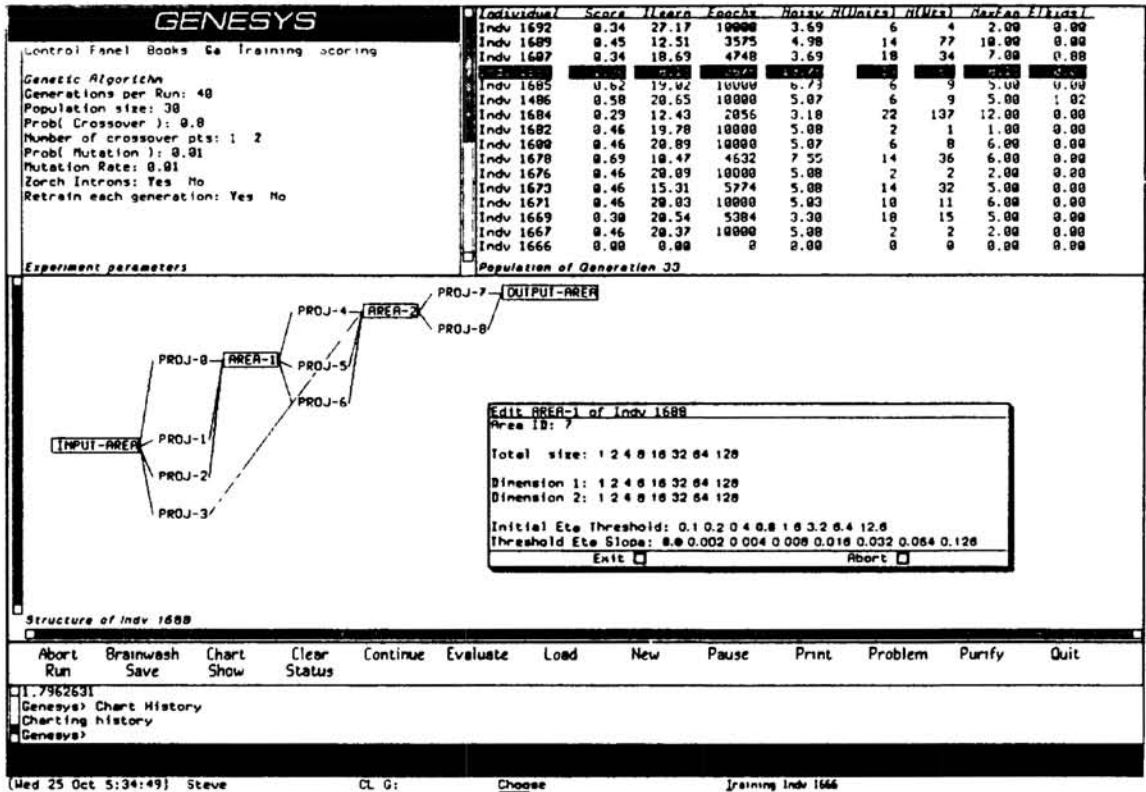

**Figure 3:** The NeuroGENESYS interface, showing a network structure optimised for the sine function problem

for program development, and NeuroGENESYS' user interface features would not have been possible without it, but the performance penalty has been severe (relative to machines with floating point hardware).

• The genetic algorithm is an inherently parallel optimization procedure, a feature we soon hope to take advantage of. We have recently implemented a networked version of NeuroGENESYS that will allow us to retain the desirable aspects of the Symbolics version and yet achieve substantial speedup in execution (we expect two to three orders of magnitude): up to 30 Apollo workstations, a VAX, and 10 Symbolics computers can now be evaluating different networks in parallel (Harp, Samad and Guha, 1990).

• The current version of NeuroGENESYS employs the backpropagation learning rule, which is notoriously slow for many applications. However, faster-learning extensions of backpropagation are continually being developed. We have incorporated one recent extension (Samad, 1988), but others, especially common ones such as including a "momentum" term in the weight update rule (Rumelhart, Hinton and Williams, 1985), could also be considered. More generally, learning in neural networks is a topic of intensive research and it is likely that more efficient learning algorithms will become popular in the near future.

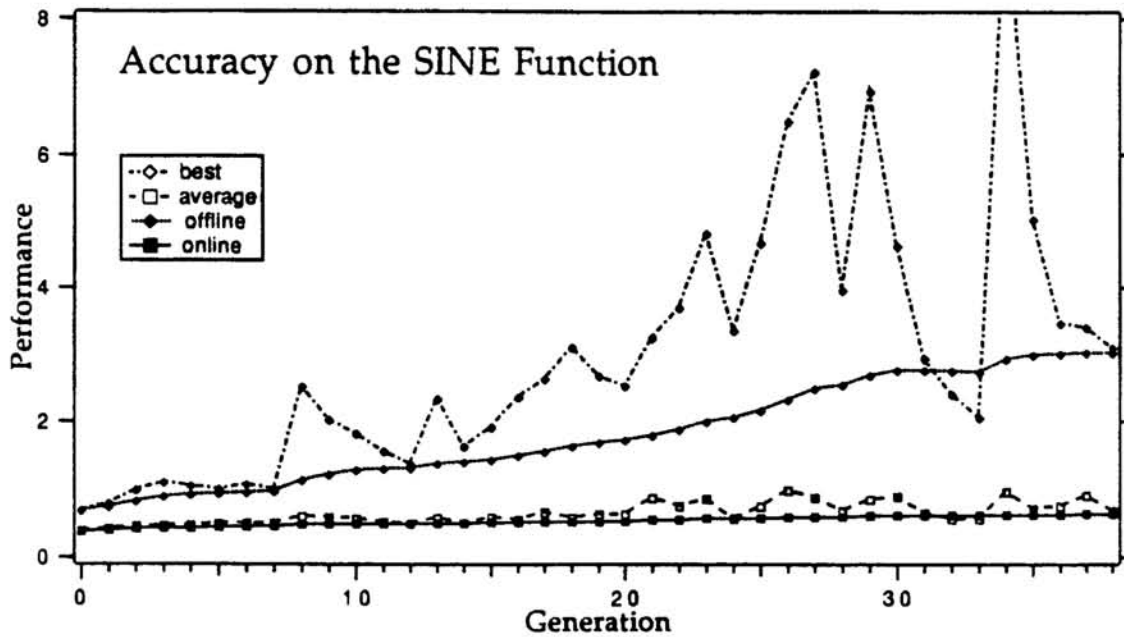

**Figure 4:  A learning curve for the sine function problem**

- The genetic algorithm is an active field of research itself. Improvements, many of which are concerned with convergence properties, are frequently being reported and could reduce the computational requirements for its application significantly.

- The genetic algorithm is an iterative optimization procedure that, on the average, produces better solutions with each passing generation. Unlike some other optimization techniques, useful results can be obtained during a run. The genetic algorithm can thus take advantage of whatever time and computational resources are available for an application.

- Just as there is no strict termination requirement for the genetic algorithm, there is no constraint on its initialization. In our experiments, the zeroth generation consisted of randomly generated networks. Not surprisingly, almost all of these are poor performers. However, better better ways of selecting the initial population are possible. In particular, the initial population can consist of manually optimized networks. Manual optimization of neural networks is currently the norm, but it leaves much of the design space unexplored. Our approach would allow a human application developer to design one or more networks that could be the starting point for further, more systematic optimization by the genetic algorithm. Other initialization approaches are also possible, such as using optimized networks from similar applications, or using heuristic guidelines to generate networks.

It should be emphasized that computational efficiency is not the only factor that must be considered in evaluating this (or any) approach. Others such as the potential for improved performance of neural network applications and the costs and benefits associated with alternative approaches for designing network applications are also critically important.

## 6  FUTURE RESEARCH

In addition to running further experiments, we hope in the future to develop versions of NeuroGENESYS for other network models, including hybrid models that incorporate supervised and unsupervised learning components.

Space restrictions have precluded a detailed description of NeuroGENESYS and our experiments. The interested reader is referred to (Harp, Samad, and Guha, 1989ab, 1990).

### References

Davis, L. (1988). Properties of a hybrid neural network-classifier system. In *Advances in Neural Information Processing Systems 1*, D.S. Touretzky (Ed.). San Mateo: Morgan Kaufmann.

Goldberg, D.E. (1989). *Genetic Algorithms in Search, Optimization and Machine Learning*. Addison-Wesley.

Harp, S.A., T. Samad, and A. Guha (1989a). Towards the genetic synthesis of neural networks. *Proceedings of the Third International Conference on Genetic Algorithms*, J.D. Schaffer (ed.). San Mateo: Morgan Kaufmann.

Harp, S.A., T. Samad, and A. Guha (1989b). *Genetic Synthesis of Neural Networks*. Technical Report I4852-CC-1989-2. Honeywell SSDC, 1000 Boone Avenue North, Golden Valley, MN 55427.

Harp, S.A., T. Samad, and A. Guha (1990). Genetic synthesis of neural network architecture. In *The Genetic Algorithms Handbook*, L.D. Davis (Ed.). New York: Van Nostrand Reinhold. (To appear.)

Holland, J. (1975). *Adaptation in Natural and Artificial Systems*. Ann Arbor: University of Michigan Press.

Rumelhart, D.E., G.E. Hinton, and R.J. Williams (1985). *Learning Internal Representations by Error-Propagation*, ICS Report 8506, Institute for Cognitive Science, UCSD, La Jolla, CA.

Samad, T. (1988). Back-propagation is significantly faster if the expected value of the source unit is used for update. *Neural Networks*, *1*, Sup. 1.

Werbos, P. (1974). *Beyond Regression: New Tools for Prediction and Analysis in the Behavioral Sciences*. Ph.D. Thesis, Harvard University Committee on Applied Mathematics, Cambridge, MA.

Whitley, D. (1988). *Applying Genetic Algorithms to Neural Net Learning*. Technical Report CS-88-128, Department of Computer Science, Colorado State University.
